# Using Combinatorial Optimization
# within Max-Product Belief Propagation

**John Duchi**      **Daniel Tarlow**      **Gal Elidan**      **Daphne Koller**
Department of Computer Science
Stanford University
Stanford, CA 94305-9010
{jduchi,dtarlow,galel,koller}@cs.stanford.edu

## Abstract

In general, the problem of computing a *maximum a posteriori (MAP)* assignment in a Markov random field (MRF) is computationally intractable. However, in certain subclasses of MRF, an optimal or close-to-optimal assignment can be found very efficiently using combinatorial optimization algorithms: certain MRFs with mutual exclusion constraints can be solved using bipartite matching, and MRFs with *regular* potentials can be solved using minimum cut methods. However, these solutions do not apply to the many MRFs that contain such tractable components as sub-networks, but also other non-complying potentials. In this paper, we present a new method, called COMPOSE, for exploiting combinatorial optimization for sub-networks within the context of a max-product belief propagation algorithm. COMPOSE uses combinatorial optimization for computing exact max-marginals for an entire sub-network; these can then be used for inference in the context of the network as a whole. We describe highly efficient methods for computing max-marginals for subnetworks corresponding both to bipartite matchings and to regular networks. We present results on both synthetic and real networks encoding correspondence problems between images, which involve both matching constraints and pairwise geometric constraints. We compare to a range of current methods, showing that the ability of COMPOSE to transmit information globally across the network leads to improved convergence, decreased running time, and higher-scoring assignments.

## 1   Introduction

Markov random fields (MRFs) [12] have been applied to a wide variety of real-world problems. However, the probabilistic inference task in MRFs — computing the posterior distribution of one or more variables — is tractable only in small tree-width networks, which are not often an appropriate model in practice. Thus, one typically must resort to the use of approximate inference methods, most commonly (in recent years) some variant of loopy belief propagation [11].

An alternative approach, whose popularity has grown in recent years, is based on the *maximum a posteriori (MAP)* inference problem — computing the single most likely assignment relative to the distribution. Somewhat surprisingly, there are certain classes of networks where MAP inference can be performed very efficiently using combinatorial optimization algorithms, even though posterior probability inference is intractable. So far, two main such classes of networks have been studied. *Regular (or associative) networks* [18], where the potentials encode a preference for adjacent variables to take the same value, can be solved optimally or almost optimally using a minimum cut algorithm. Conversely, *matching networks*, where the potentials encode a type of mutual exclusion constraints between values of adjacent variables, can be solved using matching algorithms. These types of networks have been shown to be applicable in a variety of applications, such as stereo reconstruction [13] and segmentation for regular networks, and image correspondence [15] or word alignment for matching networks [19].

In many real-world applications, however, the problem formulation does not fall neatly into one of these tractable subclasses. The problem may well have a large component that can be well-modeled as regular or as a matching problem, but there may be additional constraints that take it outside this restricted scope. For example, in a task of registering features between two images or 3D scans, we may formulate the task as a matching problem, but may also want to encode constraints that enforce the preservation of local or global geometry [1]. Unfortunately, once the network contains some "non-complying" potentials, it is not clear if and how one can apply the combinatorial optimization algorithm, even if only as a subroutine. In practice, in such networks, one often simply resorts to applying standard inference methods, such as belief propagation. Unfortunately, belief propagation may be far from an ideal procedure for these types of networks. In many cases, the MRF structures associated with the tractable components are quite dense and contain many small loops, leading to convergence problems and bad approximations. Indeed, recent empirical studies studies [17] show that belief propagation methods perform considerably worse than min-cut-based methods when applied to a variety of (purely) regular MRFs. Thus, falling back on belief propagation methods for these MRFs may result in poor performance.

The main contribution of this paper is a message-passing scheme for max-product inference that can exploit combinatorial optimization algorithms for tractable subnetworks. The basic idea in our algorithm, called COMPOSE (**C**ombinatorial **O**ptimization for **M**ax-**P**roduct **on** **S**ubnetworks), is that the network can often be partitioned into a number of subnetworks whose union is equivalent to the original distribution. If we can efficiently solve the MAP problem for each of these subnetworks, we would like to combine these results in order to find an approximate MAP for the original problem. The obvious difficulty is that a MAP solution, by itself, provides only a single assignment, and one cannot simply combine different assignments. The key insight is that we can combine the information from the different sub-networks by computing *max-marginals* for each one. A max-marginal for an individual variable $X$ is a vector that specifies, for each value $x$, the probability of the MAP assignment in which $X = x$. If we have a black box that computes a max-marginal for each variable $X$ in a subnetwork, we can embed that black box as a subroutine in a max-product belief propagation algorithm, without changing the algorithm's basic properties.

In the remainder of this paper, we define the COMPOSE scheme, and show how combinatorial algorithms for both regular networks and matching networks can be embedded in this framework. In particular, we also describe efficient combinatorial optimization algorithms for both types of networks that can compute all the max-marginals in the network at a cost similar to that of finding the single MAP assignment. We evaluate the applicability of COMPOSE on synthetic networks and on an image registration task for scans of a cell obtained using an electron microscope, all of which are matching problems with additional pairwise constraints. We compare COMPOSE to variants of both max-product and sum-product belief propagation, as well as to straight matching. Our results demonstrate that the ability of COMPOSE to transmit information globally across the network leads to improved convergence, decreased running time, and higher-scoring assignments.

## 2 Markov Random Fields

In this paper, for simplicity of presentation, we restrict our discussion to pairwise Markov networks (or Markov Random Fields) over discrete variables $\mathbf{X} = \{X_1, \ldots, X_N\}$. We emphasize that our results extend easily to the more general case of non-pairwise Markov networks. We denote an assignment of values to $\mathbf{X}$ with $\mathbf{x}$, and an assignment of a value to a single variable $X_i$ with $x_i$. A pairwise Markov network $\mathcal{M}$ is defined as a graph $\mathcal{G} = (\mathcal{V}, \mathcal{E})$ and set of potentials $\mathcal{F}$ that include both node potentials $\phi_i(x_i)$ and edge potentials $\phi_{ij}(x_i, x_j)$. The network encodes a joint probability distribution via an unnormalized density $P'_{\mathcal{F}}(\mathbf{x}) = \prod_{i=1}^{N} \phi_i(x_i) \prod_{i,j \in \mathcal{U}} \phi_{ij}(x_i, x_j)$, defining the distribution as $P_{\mathcal{F}}(\mathbf{x}) = \frac{1}{Z} P'_{\mathcal{F}}(\mathbf{x})$, where $Z$ is the *partition function* given by $Z = \sum_{\mathbf{x}'} P'_{\mathcal{F}}(\mathbf{x})$.

There are different types of queries that one may want to compute on a Markov network. Most common are *(conditional) probability queries*, where the task is to compute the marginal probability of one or more variables, possibly given some evidence. This type of inference is essentially equivalent to computing the partition function, which sums up exponentially many assignments, a computation which is currently intractable except in networks of low *tree width*. An alternative type of inference task is the is *maximum a posteriori* (MAP) problem — finding $\arg\max_{\mathbf{x}} P_{\mathcal{F}}(\mathbf{x}) = \arg\max_{\mathbf{x}} P'_{\mathcal{F}}(\mathbf{x})$. In the MAP problem, we can avoid computing the partition function, so there are certain classes of networks to which the MAP assignment can be computed effectively, even though computing the partition problem can be shown to be intractable; we describe two such important classes in Section 4.

In general, however, an exact solution to the MAP problem is also intractable. *Max-product belief propagation (MPBP)* [20] is a commonly-used method for finding an approximate solution. In this algorithm, each node $X_i$ passes to its neighboring nodes $\mathcal{N}_i$ a message which is a vector defining a value for each value $x_i$:

$$\delta_{i \to j}(x_j) := \max_{x_i} \left[ \phi_i(x_i)\phi_{ij}(x_i, x_j) \prod_{k \in \mathcal{N}_i - \{j\}} \delta_{k \to i}(x_i) \right].$$

At convergence, each variable can compute its own local belief as: $b_i(x_i) = \phi_i(x_i) \prod_{k \in \mathcal{N}_i} \delta_{k \to i}(x_i)$. In a tree structured MRF, if such messages are passed from the leaves towards a single root, the value of the message passed by $X_i$ towards the root encodes a partial *max-marginal*: the entry for $x_i$ is the probability of the most likely assignment, to the subnetwork emanating from $X_i$ away from the root, where we force $X_i = x_i$. At the root, we obtain exact max-marginals for the entire joint distribution. However, applied to a network with loops, MPBP often does not converge, even when combined with techniques such as smoothing and asynchronous message passing, and the answers obtained can be quite approximate.

## 3   Composing Max-Product Inference on Subnetworks

We now describe the COMPOSE scheme for decomposing the network into hopefully more tractable components, and allowing approximate max-product computation over the network as a whole to be performed by iteratively computing max-product in one component and passing approximate max-marginals to the other(s). As the unnormalized probability of an assignment in a Markov network is a product of local potentials, we can partition the potentials in an MRF into an ensemble of $k$ subgraphs $\mathcal{G}_1, \ldots \mathcal{G}_k$ over the same set of nodes $\mathcal{V}$, associated edges $\mathcal{E}_1, \ldots, \mathcal{E}_k$ and sets of factors $\mathcal{F}_1, \ldots, \mathcal{F}_k$. We require that the product of the potentials in these subnetworks maintain the same information as the original MRF. That is, if we originally have a factor $\phi_i \in \mathcal{F}$ and associated factors $\phi_i^{(1)} \in \mathcal{F}_1, \ldots, \phi_i^{(k)} \in \mathcal{F}_k$, we must have that $\prod_{l=1}^{k} \phi_i^{(l)}(X_i) = \phi_i(X_i)$. One method of partitioning that achieves this equality is simply to select, for each potential $\phi_i$, one subgraph in which it appears unchanged, and set all of the other $\phi_i^{(l)}$ to be $1$.

Even if MAP inference in the original network is intractable, it may be tractable in each of the sub-networks in the ensemble. But how do we combine the results from MAP inference in an ensemble of networks over the same set of variables? Our approach draws its motivation from the MPBP algorithm, which computes messages that correspond to pseudo-max-marginals over single variables (approximate max-marginals, that do not account for the loops in the network). We begin by conceptually reformulating the ensemble as a set of networks over disjoint sets of variables $\{X_1^{(l)}, \ldots, X_n^{(l)}\}$ for $l = 1, \ldots, k$; we enforce consistency of the joint assignment using a set of "communicator" variables $X_1, \ldots, X_n$, such that each $X_i^{(l)}$ must take the same value as $X_i$. We assume that each subnetwork is associated with an algorithm that can "read in" pseudo-max-marginals over the communicator variables, and compute pseudo-max-marginals over these variables.

More precisely, let $\delta_{(l) \to i}$ be the message sent from subnetwork $l$ to $X_i$ and $\delta_{i \to (l)}$ the opposite message. Then we define the COMPOSE message passing scheme as follows:

$$\delta_{(l) \to i}(x_i) = \max_{\mathbf{x}^{(l)} : X_i^{(l)} = x_i} P_{\mathcal{F}_l}(\mathbf{x}^{(l)}) \prod_{j \neq i} \delta_{j \to (l)}(X_j^{(l)}) \qquad (1)$$

$$\delta_{i \to (l)} = \prod_{l' \neq l} \delta_{(l') \to i}. \qquad (2)$$

That is, each subnetwork computes its local pseudo-max-marginals over each of the individual variables, given, as input, the pseudo-max-marginals over the others. The separate pseudo-max-marginals are integrated via the communicator variables. It is not difficult to see that this message passing scheme is equivalent to a particular scheduling algorithm for max-product belief propagation over the ensemble of networks, assuming that the max-product computation in each of the subnetworks is computed exactly using a black-box subroutine.

We note that this message passing scheme is somewhat related to the *tree-reweighted max-product (TRW)* method of Wainwright et al. [8], where the network distribution is partitioned as a weighted combination of trees, which also communicate pseudo-max-marginals with each other.

# 4 Efficient Computation of Max-Marginals

In this section, we describe two important classes of networks where the MAP problem can be solved efficiently using combinatorial algorithms: matching networks, which can be solved using bipartite matching algorithms; and regular networks, which can be solved using (iterated application of) minimum cut algorithms. We show how the same algorithms can be adapted, at minimal computational cost, for computing not only the single MAP assignment, but also the set of max-marginals. This allows these algorithms to be used as one of our "black boxes" in the COMPOSE framework.

**Bipartite matching.** Many problems can be well-formulated as maximum-score (or minimum weight) bipartite matching: We are given a graph $\mathcal{G} = (\mathcal{A}, \mathcal{U})$, whose nodes are partitioned into disjoint sets $\mathcal{A} = A \cup B$. In $\mathcal{G}$, each edge $(a, b)$ has one endpoint in $A$ and the other in $B$ and an associated score $c(a, b)$. A bipartite matching is a subset of the edges $\mathcal{W} \subset \mathcal{U}$ such that each node appears in at most one edge. The notion of a matching can be relaxed to include other types of degree constraints, e.g., constraining certain nodes to appear in at most $k$ edges. The score of the matching is simply the sum of the scores of the edges in $\mathcal{W}$.

The matching problem can also be formulated as an MRF, in several different ways. For example, in the degree-1 case (each node in $A$ is matched to one node in $B$), we can have a variable $X_a$ for each $a \in A$ whose possible values are all of the nodes in $B$. The edge scores in the matching graph are then simply singleton potentials in the MRF, where $\phi_a(X_a = b) = \exp(c(a, b))$. Unfortunately, while the costs can be easily encoded in an MRF, the degree constraints on the matching induce a set of pairwise mutual-exclusion potentials on all pairs of variables in the MRF, leading to a fully connected network. Thus, standard methods for MRF inference cannot handle the networks associated with matching problems.

Nevertheless, finding the maximum score bipartite matching (with any set of degree constraints) can be accomplished easily using standard combinatorial optimization algorithms (e.g., [6]). However, we also need to find all the max-marginals. Fortunately, we can adapt the standard algorithm for finding a single best matching to also find all of the max-marginals. A standard solution to the max-matching problem reduces it to a max-weight flow problem, by introducing an additional "source" node that connects to all the nodes in $A$, and an additional "sink" node that connects to all the nodes in $B$; the capacity of these edges is the degree constraint of the node (1 for a 1-to-1 matching). We now run a standard max-weight flow algorithm, and define an edge to be in the matching if it bears flow. Standard results show that, if the edge capacities are integers, then the flow too is integral, so that it defines a matching. Let $w^*$ be the weight of the flow in the graph. A flow in the graph defines a *residual graph*, where there is an edge in the graph whose capacity is the amount of flow it can carry *relative to the current flow*. Thus, for example, if the current solution carries a unit of flow along a particular edge $(a, b)$ in the original graph, the residual graph will have an edge with a unit capacity going in the reverse direction, corresponding to the fact that we can now choose to "eliminate" the flow from $a$ to $b$. The scores in these inverse edges are also negative, corresponding to the fact that score is lost when we reduce the flow.

Our goal now is to find, for each pair $(a, b)$, the score of the optimal matching where we force this pair to be matched. If this pair is matched in the current solution, then the score is simply $w^*$. Otherwise, we simply find the highest scoring path from $b$ to $a$ in the residual graph. Any edges on this new path from $A$ to $B$ will be included in the new matching; any edges from $B$ to $A$ were included in the old matching, but are not in the new matching because of the augmenting path. This path is the best way of changing the flow so as to force flow from $a$ to $b$. Letting $\Delta$ be the weight of this augmenting path, the overall score of the new flow is $w^* + \Delta$. It follows that the cost of this path is necessarily negative, for otherwise it would have been optimal to apply it to the original flow, improving its score. Thus, we can find the highest-scoring path by simply negating all edge costs and finding the shortest path in the graph.

Thus, to compute all of the max-marginals, we simply need to find the shortest path from every node $a \in A$ to every node $b \in B$. We can find this using the Floyd-Warshall all-pairs-shortest-paths algorithm, which runs in $O((n_A + n_B)^3)$ time, for $n_A = |A|$ and $n_B = |B|$; or we can run a single-source shortest-path algorithm for each node in $B$, at a total cost of $O(n_B \cdot n_A n_B \log(n_A n_B))$. By comparison, the cost of solving the initial flow problem is $O(n_A^3 \log(n_A))$.

**Minimum Cuts.** A very different class of networks that admits an efficient solution is based on the application of a *minimum cut* algorithm to a graph. At a high level, these networks encode situations where adjacent variables like to take "similar" values. There are many variants of this condition. The simplest variant is applied to pairwise MRFs over binary-valued random variables.

In this case, a potential is said to be *regular* if: $\phi_{ij}(X_i = 1, X_j = 1) \cdot \phi_{ij}(X_i = 0, X_j = 0) \geq \phi_{ij}(X_i = 0, X_j = 1) \cdot \phi_{ij}(X_i = 1, X_j = 0)$. For MRFs with only regular potentials, the MAP solution can be found as the minimum cut of a weighted graph constructed from the MRF [9]. This construction can be extended in various ways (see [9] for a survey), including to the class of networks with non-binary variables whose negative-log-probability is a convex function [5]. Moreover, for a range of conditions on the potentials, an $\alpha$-*expansion* procedure [2], which iteratively applies a min-cut to a series of graphs, can be used to find a solution with guaranteed approximation error relative to the optimal MAP assignment.

As above, a single joint assignment does not suffice for our purposes. In recent work, Kohli and Torr [7], studying the problem of confidence estimation in MAP problems, showed how all of the max-marginals in a regular network can be computed using dynamic algorithms for flow computations. Their method also applies to non-binary networks with convex potentials (as in [5]), but not to networks for which $\alpha$-expansion is used to find an approximate MAP assignment.

## 5  Experimental Results

We evaluate COMPOSE on the image correspondence problem, which is characteristic of matching problems with geometric constraints. We compare both max-product tree-reparameterization (**TRMP**) [8] and asynchronous max-product (**AMP**). The axes along which we compare all algorithms are: the ability to achieve convergence, the time it takes to reach a solution, and the quality — log of the unnormalized likelihood — of the solution found, in the Markov network that defines the problem. We use standard message damping of .3 for the max-product algorithms and a convergence threshold of $10^{-3}$ for all propagation algorithms. All tests were run on a 3.4 GHz Pentium 4 processor with 2GB of memory.

We focus our experiments on an image correspondence task, where the goal is to find a 1-to-1 mapping between landmarks in two images. Here, we have a set of template points $\mathcal{S} = \{\mathbf{x}_1, \ldots, \mathbf{x}_n\}$ and a set $\mathcal{T}$ of target points, $\{\mathbf{x}'_1, \ldots, \mathbf{x}'_n\}$. We encode our MRF with a variable $X_i$ for each marker $\mathbf{x}_i$ in the source image, whose value corresponds to its aligned candidate $\mathbf{x}'_j$ in the target image. Our MRF contains singleton potentials $\phi_i$, which may encode both local appearance information, so that a marker $\mathbf{x}_i$ prefers to be aligned to a candidate $\mathbf{x}'_j$ in the target image whose neighborhood looks similar to $\mathbf{x}_i$'s, or a distance potential so that markers $\mathbf{x}_i$ prefer to be aligned to candidates $\mathbf{x}'_j$ in locations close to those in the source image. The MRF also contains pairwise potentials $\{\phi_{ij}\}$ that can encode dependencies between the landmark assignments. In particular, we may want to encode geometric potentials, which enforce a preference for preservation of distance or orientation for pairs of markers $\mathbf{x}_i, \mathbf{x}_j$ and their assigned targets $\mathbf{x}'_k, \mathbf{x}'_l$. Finally, as the goal is to find a 1-to-1 mapping between landmarks in the source and target images, we also encode a set of mutual exclusion potentials over pairs of variables, enforcing the constraint that no two markers are assigned to the same candidate $\mathbf{x}'_k$. Our task is to find the MAP solution in this MRF.

**Synthetic Networks.**   We first experimented with synthetically generated networks that follow the above form. To generate the networks, we first create a source "image" that contains a set of template points $\mathcal{S} = \{\mathbf{x}_1, \ldots, \mathbf{x}_n\}$, chosen by uniformly sampling locations from a two-dimensional plane. Next, the target set of points $\mathcal{T} = \{\mathbf{x}'_1, \ldots, \mathbf{x}'_n\}$ is generated by generating one point from each template point $\mathbf{x}_i$, sampling from a Gaussian distribution with mean $\mathbf{x}_i$ and a diagonal covariance matrix $\sigma^2 \mathbf{I}$. As there was no true local information, the matching (or singleton) potentials for both types of synthetic networks were generated uniformly at random on $[0, 1)$. The 'correct' matching point, or the one the template variable generates, was given weight .7, ensuring that the correct matching gets a non-negligible weight without making the correspondence too obvious. We consider two different formulations for the geometric potentials. The first utilizes a minimum spanning tree connecting the points in $\mathcal{S}$, and the second simply a chain. In both cases, we generate pairwise geometric potentials $\phi_{ij}(X_i, X_j)$ that are Gaussian with mean $\mu = (\mathbf{x}_i - \mathbf{x}_j)$ and standard deviation proportional to the Euclidean distance between $\mathbf{x}_i$ and $\mathbf{x}_j$ and variance $\sigma^2$. Results for the two constructions were similar, so, due to lack of space, we present results only for the line networks.

Fig. 1(a) shows the cumulative percentage of convergent runs as a function of CPU time. COMPOSE converges significantly more often than either AMP or state-of-the-art TRMP. For TRMP, we created one tree over all the geometric and singleton potentials to quickly pass information through the entire graph; the rest of the trees chosen for TRMP were over a singleton potential, all the neighboring mutual exclusion potentials, and pairwise potentials neighboring the singleton, allowing us to maintain the mutual exclusion constraints during different reparameterization steps in TRMP. Since

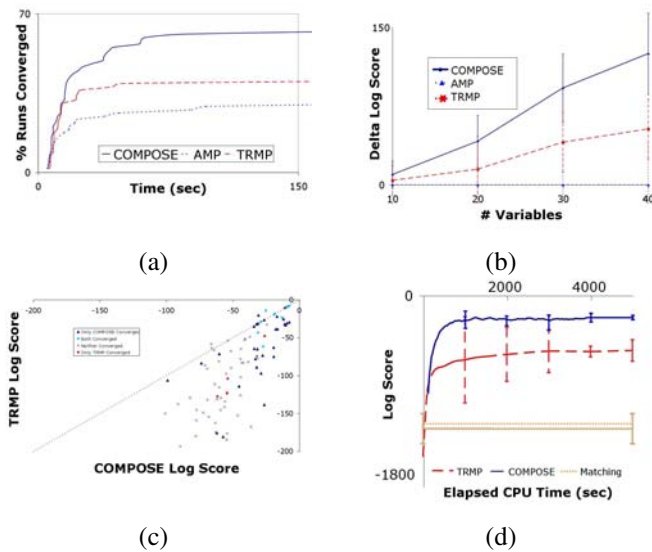

Figure 1: (a) Cumulative percentage of convergent runs versus CPU time on networks with 30 variables and sigma ranging from 3 to 9. (b) The effect of changing the number of variables on the log score. Shown is the difference between the log score of each algorithm and the score found by AMP. (c) Direct comparison of COMPOSE to TRMP on individual runs from the same set of networks as in (b), grouped by algorithm convergence. (d) Score of assignment based on intermediate beliefs versus time for COMPOSE, TRMP, and matching on 100 variable networks. All algorithms were allowed to run for 5000 seconds.

sum-product algorithms are known in general to be less susceptible to oscillation than their max-product counterparts, we also compared against sum-product asynchronous belief propagation. In our experiments, however, sum-production BP did not achieve good scores even on runs in which it did converge, perhaps because the distribution was fairly diffuse, leading to an averaging of diverse solutions; we omit results for lack of space.

Fig. 1(b) shows the average difference in log scores between each algorithm's result and the average log score of AMP as a function of the number of variables in the networks. COMPOSE clearly outperforms the other algorithms, gaining a larger score margin as the size of the problem increases. In the synthetic tests we ran for (b) and (c), COMPOSE achieved the best score in over 90% of cases. This difference was greatest in more difficult problems, where there is greater variance in the locations of candidates in the target image leading to difficulty achieving a 1-to-1 correspondence.

In Fig. 1(c), we further examine scores from individual runs, comparing COMPOSE directly to the strongest competitor, TRMP. COMPOSE consistently outperforms TRMP and never loses by more than a small margin; COMPOSE often achieves scores on the order of $2^{40}$ times better than those achieved by TRMP. Interestingly, there appears not to be a strong correlation between relative performance and whether or not the algorithms converged.

Fig. 1(d) examines the intermediate scores obtained by COMPOSE and TRMP on intermediate assignments reached during the inference process, for large (100 variable) problems. Though COMPOSE does not reach convergence in messages, it quickly takes large steps to a very good score on the large networks. TRMP also takes larger steps near the beginning, but it is less consistent and it never achieves a score as high as COMPOSE. This indicates that COMPOSE scales better than TRMP to larger problems. This behavior may also help to explain the results from (c), where we see that, even when COMPOSE does not converge in messages, it still is able to achieve good scores. Overall, these results indicate that we can use intermediate results for COMPOSE even before convergence.

**Real Networks.** We now consider real networks generated for the task of electron microscope tomography: the three-dimensional reconstruction of cell and organelle structures based on a series of images obtained at different tilt angles. The problem is to localize and track markers in images across time, and it is a difficult one; traditional methods like cross correlation and graph matching often result in many errors. We can encode the problem, however, as an MRF, as described above. In this case, the geometric constraints were more elaborate, and it was not clear how to construct a good set of spanning trees. We therefore used a variant on AMP called residual max-product (**RMP**) [3] that schedules messages in an informed way over the network; in this work and others, we have found this variant to achieve better performance than TRMP on difficult networks.

Fig. 2(a) shows a source set of markers in an electron tomography image; Fig. 2(b) shows the correspondence our algorithm achieves, and Fig. 2(c) shows the correspondence that RMP achieves. Note that, in Fig. 2(c), points from the source image are assigned to the same point in the target image, whereas COMPOSE does not have the same failing. Of the twelve pairs of images we tested,

RMP failed to converge on 11/12 within 20 minutes, whereas COMPOSE failed to converge on only two of the twelve. Because the network structure was difficult for loopy approximate methods, we ran experiments where we replaced mutual exclusion constraints with soft location constraints on individual landmarks; while convergence improved, actual performance was inferior.

Fig. 2(d) shows the scores for the different methods we use to solve these problems. Using RMP as the baseline score, we see the difference in scores for the different methods. It is clear that, though RMP and TRMP run on a simpler network with soft mutual exclusion constraints are competitive with, and even very slightly better than COMPOSE on simple problems, as problems become more difficult (more variance in target images), COMPOSE clearly dominates. We also compare COMPOSE to simply finding the best matching of markers to candidates without any geometric information; COMPOSE dominates this approach, never scoring worse than the matching.

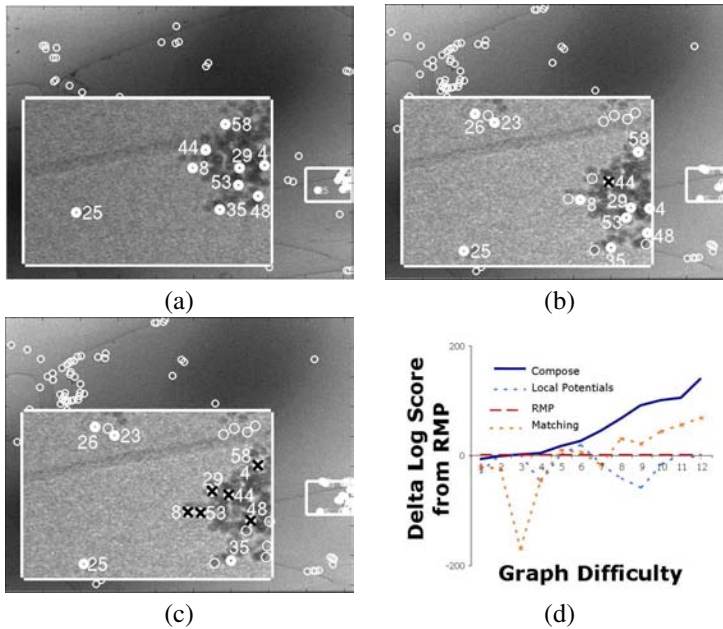

(a)  (b)

(c)  (d)

Figure 2: (a) Labeled markers in a source electron microscope image (b) Candidates COMPOSE assigns in the target image (c) Candidates RMP assigns in the target image (note the Xs through incorrect or duplicate assignments) (d) A score comparison of COMPOSE, matching, and RMP on the image correspondences

## 6  Discussion

In this paper, we have presented COMPOSE, an algorithm that exploits the presence of tractable substructures in MRFs within the context of max-product belief propagation. Motivated by the existence of very efficient algorithms to extract all max-marginals from combinatorial substructures, we presented a variation of belief propagation methods that used the max-marginals to take large steps in inference. We also demonstrated that COMPOSE significantly outperforms state-of-the-art methods on different challenging synthetic and real problems.

We believe that one of the major reasons that belief propagation algorithms have difficulty with the augmented matching problems described above is that the mutual exclusion constraints create a phenomenon where small changes to local regions of the network can have strong effects on distant parts of the network, and it is difficult for belief propagation to adequately propagate information. Some existing variants of belief propagation (such as TRMP) attempt to speed the exchange of information across opposing sides of the network by means of intelligent message scheduling. Even intelligently-scheduled message passing is limited, however, as messages are inherently local. If there are oscillations across a wide diameter, due to global interactions in the network, they might contribute significantly to poor performance by BP algorithms.

COMPOSE slices the network along a different axis, using subnetworks that are global in nature but that do not have all of the information about any subset of variables. If the component of the network that is difficult for belief propagation can be encoded in an efficient special-purpose subnetwork such as a matching, then we have a means of effectively propagating global information. We conjecture that COMPOSE's ability to globally pass information contributes both to its improved convergence and to the better results it obtains even without convergence.

Some very recent work explores the case where a regular MRF contains terms that are not regular [14, 13], but this work is largely specific to certain types of "close-to-regular" MRFs. It would be interesting to compare COMPOSE and these methods on a range of networks containing regu-

lar subgraphs. Our work is also related to work trying to solve the *quadratic assignment problem (QAP)* [10], a class of problems of which our generalized matching networks are a special case. Standard algorithms for QAP include simulated annealing, tabu search, branch and bound, and *ant algorithms* [16]; the latter have some of the flavor of message passing, walking trails over the graph representing a QAP and iteratively updating scores of different assignments to the QAP. To the best of our knowledge, however, none of these previous methods attempts to use a combinatorial algorithm as a component in a general message-passing algorithm, thereby exploiting the structure of the pairwise constraints.

There are many interesting directions arising from this work. It would be interesting to perform a theoretical analysis of the COMPOSE approach, perhaps providing conditions under which it is guaranteed to provide a certain level of approximation. A second major direction is the identification of other tractable components within real-world MRFs that one can solve using combinatorial optimization methods, or other efficient approaches. For example, the constraint satisfaction community has studied several special-purpose constraint types that can be solved more efficiently than using generic methods [4]; it would be interesting to explore whether these constraints arise within MRFs, and, if so, whether the special-purpose procedures can be integrated into the COMPOSE framework. Overall, we believe that real-world MRFs often contain large structured sub-parts that can be solved efficiently with special-purpose algorithms; the combination of special-purpose solvers within a general inference scheme may allow us to solve problems that are intractable to any current method.

## Acknowledgments

This research was supported by the Defense Advanced Research Projects Agency (DARPA) under the Transfer Learning Program. We also thank David Karger for useful conversations and insights.

## References

[1] D. Anguelov, D. Koller, P. Srinivasan, S. Thrun, H. Pang, and J. Davis. The correlated correspondence algorithm for unsupervised registration of nonrigid surfaces. In *NIPS*, 2004.

[2] Y. Boykov, O. Veksler, and R. Zabih. Fast approximate energy minimization via graph cuts. In *ICCV*, 1999.

[3] G. Elidan, I. McGraw, and D. Koller. Residual belief propagation. In *UAI*, 2006.

[4] J. Hooker, G. Ottosson, E.S. Thorsteinsson, and H.J. Kim. A scheme for unifying optimization and constraint satisfaction methods. In *Knowledge Engineering Review*, 2000.

[5] H. Ishikawa. Exact optimization for Markov random fields with convex priors. *PAMI*, 2003.

[6] J. Kleinberg and E. Tardos. *Algorithm Design*. Addison-Wesley, 2005.

[7] P. Kohli and P. Torr. Measuring uncertainty in graph cut solutions - efficiently computing min-marginal energies using dynamic graph cuts. In *ECCV*, 2006.

[8] V. Kolmogorov and M. Wainwright. On the optimality of tree-reweighted max-product message-passing. In *UAI '05*.

[9] V. Kolmogorov and R. Zabih. What energy functions can be minimized via graph cuts? In *ECCV*, 2002.

[10] E. Lawler. The quadratic assignment problem. In *Management Science*, 1963.

[11] K. Murphy and Y. Weiss. Loopy belief propagation for approximate inference: An empirical study. In *UAI '99*.

[12] J. Pearl. *Probabilistic Reasoning in Intelligent Systems*. Morgan Kaufmann, 1988.

[13] A. Raj, G. Singh, and R. Zabih. MRF's for MRI's: Bayesian reconstruction of MR images via graph cuts. In *CVPR*, 2006. To appear.

[14] C. Rother, S. Kumar, V. Kolmogorov, and A. Blake. Digital tapestry. In *CVPR*, 2005.

[15] J. Shi and J. Malik. Normalized cuts and image segmentation. *PAMI*, 2000.

[16] T. Stützle and M. Dorigo. ACO algorithms for the quadratic assignment problem. In *New Ideas in Optimization*. 1999.

[17] R. Szeliski, R. Zabih, D. Scharstein, O. Veksler, V. Kolmogorov, A. Agarwala, M. Tappen, and C. Rother. A comparative study of energy minimization methods for Markov random fields. In *ECCV*, 2006.

[18] B. Taskar, V. Chatalbashev, and D. Koller. Learning associative markov networks. In *ICML '04*.

[19] B. Taskar, V. Chatalbashev, D. Koller, and C. Guestrin. Learning structured prediction models: a large margin approach. In *ICML '05*.

[20] Y. Weiss and W. Freeman. On the optimality of solutions of the max-product belief-propagation algorithm in arbitrary graphs. *IEEE Transactions on Information Theory*, 47, 2001.
